# Neural Implementation of Hierarchical Bayesian Inference by Importance Sampling

**Lei Shi**
Helen Wills Neuroscience Institute
University of California, Berkeley
Berkeley, CA 94720
lshi@berkeley.edu

**Thomas L. Griffiths**
Department of Psychology
University of California, Berkeley
Berkeley, CA 94720
tom_griffiths@berkeley.edu

## Abstract

The goal of perception is to infer the hidden states in the hierarchical process by which sensory data are generated. Human behavior is consistent with the optimal statistical solution to this problem in many tasks, including cue combination and orientation detection. Understanding the neural mechanisms underlying this behavior is of particular importance, since probabilistic computations are notoriously challenging. Here we propose a simple mechanism for Bayesian inference which involves averaging over a few feature detection neurons which fire at a rate determined by their similarity to a sensory stimulus. This mechanism is based on a Monte Carlo method known as importance sampling, commonly used in computer science and statistics. Moreover, a simple extension to recursive importance sampling can be used to perform hierarchical Bayesian inference. We identify a scheme for implementing importance sampling with spiking neurons, and show that this scheme can account for human behavior in cue combination and the oblique effect.

## 1   Introduction

Living creatures occupy an environment full of uncertainty due to noisy sensory inputs, incomplete observations, and hidden variables. One of the goals of the nervous system is to infer the states of the world given these limited data and make decisions accordingly. This task involves combining prior knowledge with current data [1], and integrating cues from multiple sensory modalities [2]. Studies of human psychophysics and animal behavior suggest that the brain is capable of solving these problems in a way that is consistent with optimal Bayesian statistical inference [1, 2, 3, 4]. Moreover, complex brain functions such as visual information processing involves multiple brain areas [5]. Hierarchical Bayesian inference has been proposed as a computational framework for modeling such processes [6]. Identifying neural mechanisms that could support hierarchical Bayesian inference is important, since probabilistic computations can be extremely challenging. Just representing and updating distributions over large numbers of hypotheses is computationally expensive.

Much effort has recently been devoted towards proposing possible mechanisms based on known neuronal properties. One prominent approach to explaining how the brain uses population activities for probabilistic computations has been done in the "Bayesian decoding" framework [7]. In this framework, it is assumed that the firing rate of a population of neurons, $r$, can be converted to a probability distribution over stimuli, $p(s|r)$, by applying Bayesian inference, where the likelihood $p(r|s)$ reflects the probability of that firing pattern given the stimulus $s$. A firing pattern thus encodes a distribution over stimuli, which can be recovered through Bayesian decoding. The problem of performing probabilistic computations then reduces to identifying a set of operations on firing rates $r$ that result in probabilistically correct operations on the resulting distributions $p(s|r)$. For example,

[8] showed that when the likelihood $p(r|s)$ is an exponential family distribution with linear sufficient statistics, adding two sets of firing rates is equivalent to multiplying probability distributions.

In this paper, we take a different approach, allowing a population of neurons to encode a probability distribution directly. Rather than relying on a separate decoding operation, we assume that the activity of each neuron translates directly to the weight given to the optimal stimulus for that neuron in the corresponding probability distribution. We show how this scheme can be used to perform Bayesian inference, and how simple extensions of this basic idea make it possible to combine sources of information and to propagate uncertainty through multiple layers of random variables. In particular, we focus on one Monte Carlo method, namely importance sampling with the prior as a surrogate, and show how recursive importance sampling approximates hierarchical Bayesian inference.

## 2 Bayesian inference and importance sampling

Given a noisy observation $x$, we can recover the true stimulus $x^*$ by using Bayes' rule to compute the posterior distribution

$$p(x^*|x) = \frac{p(x^*)p(x|x^*)}{\int_{x^*} p(x^*)p(x|x^*)dx^*} \tag{1}$$

where $p(x^*)$ is the prior distribution over stimulus values, and $p(x|x^*)$ is the likelihood, indicating the probability of the observation $x$ if the true stimulus value is $x^*$. A good guess for the value of $x^*$ is the expectation of $x^*$ given $x$. In general, we are often interested in the expectation of some function $f(x^*)$ over the posterior distribution $p(x^*|x)$, $E[f(x^*)|x]$. The choice of $f(x^*)$ depends on the task. For example, in noise reduction where $x^*$ itself is of interest, we can take $f(x^*) = x^*$.

However, evaluating expectations over the posterior distribution can be challenging: it requires computing a posterior distribution and often a multidimensional integration. The expectation $E[f(x^*)|x]$ can be approximated using a Monte Carlo method known as importance sampling. In its general form, importance sampling approximates the expectation by using a set of samples from some surrogate distribution $q(x^*)$ and assigning those samples weights proportional to the ratio $p(x^*|x)/q(x^*)$.

$$E[f(x^*)|x] = \int f(x^*)\frac{p(x^*|x)}{q(x^*)}q(x^*)dx^* \simeq \frac{1}{M}\sum_{i=1}^{M} f(x_i^*)\frac{p(x_i^*|x)}{q(x_i^*)} \qquad x_i^* \sim q(x^*) \tag{2}$$

If we choose $q(x^*)$ to be the prior $p(x^*)$, the weights reduce to the likelihood $p(x|x^*)$, giving

$$
\begin{aligned}
E[f(x^*)|x] &\simeq \frac{1}{M}\sum_{i=1}^{M} f(x_i^*)\frac{p(x_i^*|x)}{p(x_i^*)} = \frac{1}{M}\sum_{i=1}^{M} f(x_i^*)\frac{p(x,x_i^*)}{p(x_i^*)p(x)} = \frac{1}{M}\sum_{i=1}^{M} f(x_i^*)\frac{p(x|x_i^*)}{p(x)} \\
&= \frac{\frac{1}{M}\sum_{i=1}^{M} f(x_i^*)p(x|x_i^*)}{\int p(x|x^*)p(x^*)dx^*} \simeq \sum_{x_i^*} f(x_i^*)\frac{p(x|x_i^*)}{\sum_{x_i^*} p(x|x_i^*)} \qquad x_i^* \sim p(x^*) \tag{3}
\end{aligned}
$$

Thus, importance sampling provides a simple and efficient way to perform Bayesian inference, approximating the posterior distribution with samples from the prior weighted by the likelihood. Recent work also has suggested that importance sampling might provide a psychological mechanism for performing probabilistic inference, drawing on its connection to exemplar models [9].

## 3 Possible neural implementations of importance sampling

The key components of an importance sampler can be realized in the brain if: 1) there are feature detection neurons with preferred stimulus tuning curves proportional to the likelihood $p(x|x_i^*)$; 2) the frequency of these feature detection neurons is determined by the prior $p(x^*)$; and 3) divisive normalization can be realized by some biological mechanism. In this section, we first describe a radial basis function network implementing importance sampling, then discuss the feasibility of three assumptions mentioned above. The model is then extended to networks of spiking neurons.

### 3.1 Radial basis function (RBF) networks

Radial basis function (RBF) networks are a multi-layer neural network architecture in which the hidden units are parameterized by locations in a latent space $x_i^*$. On presentation of a stimulus $x$,

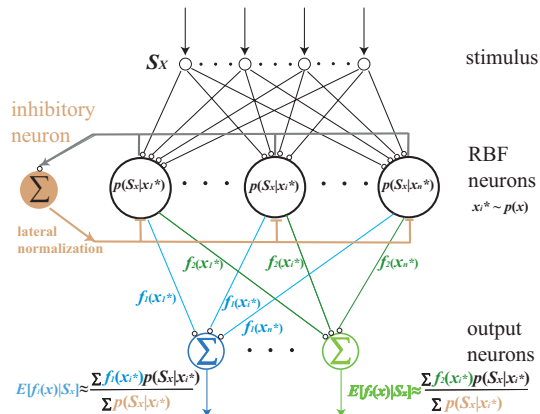

Figure 1: Importance sampler realized by radial basis function network. For details see Section 3.1.

these hidden units are activated according to a function that depends only on the distance $||x - x_i^*||$, e.g., $\exp(-|x - x_i^*|^2/2\sigma^2)$, similar to the tuning curve of a neuron. RBF networks are popular because they have a simple structure with a clear interpretation and are easy to train. Using RBF networks to model the brain is not a new idea – similar models have been proposed for pattern recognition [10] and as psychological accounts of human category learning [11].

Implementing importance sampling with RBF networks is straightforward. A RBF neuron is recruited for a stimulus value $x_i^*$ drawn from the prior (Fig. 1). The neuron's synapses are organized so that its tuning curve is proportional to $p(x|x_i^*)$. For a Gaussian likelihood, the peak firing rate would be reached at preferred stimulus $x = x_i^*$ and diminishes as $||x - x_i^*||$ increases. The $i$th RBF neuron makes a synaptic connection to output neuron $j$ with strength $f_j(x_i^*)$, where $f_j$ is a function of interest. The output units also receive input from an inhibitory neuron that sums over all RBF neurons' activities. Such an RBF network produces output exactly in the form of Eq. 3, with the activation of the output units corresponding to $E[f_j(x^*)|x]$.

Training RBF networks is practical for neural implementation. Unlike the multi-layer perceptron that usually requires global training of the weights, RBF networks are typically trained in two stages. First, the radial basis functions are determined using unsupervised learning, and then, weights to the outputs are learned using supervised methods. The first stage is even easier in our formulation, because RBF neurons simply represent samples from the prior, independent of the second stage later in development. Moreover, the performance of RBF networks is relatively insensitive to the precise form of the radial basis functions [12], providing some robustness to differences between the Bayesian likelihood $p(x|x_i^*)$ and the activation function in the network. RBF networks also produce sparse coding, because localized radial basis likelihood functions mean only a few units will be significantly activated for a given input $x$.

### 3.2 Tuning curves, priors and divisive normalization

We now examine the neural correlates of the three components in RBF model. First, responses of cortical neurons to stimuli are often characterized by receptive fields and tuning curves, where receptive fields specify the domain within a stimulus feature space that modify neuron's response and tuning curves detail how neuron's responses change with different feature values. A typical tuning curve (like orientation tuning in V1 simple cells) has a bell-shape that peaks at the neuron's preferred stimulus parameter and diminishes as parameter diverges. These neurons are effectively measure the likelihood $p(x|x_i^*)$, where $x_i^*$ is the preferred stimulus.

Second, importance sampling requires neurons with preferred stimuli $x_i^*$ to appear with frequency proportional to the prior distribution $p(x^*)$. This can be realized if the number of neurons representing $x^*$ is roughly proportional to $p(x^*)$. While systematic study of distribution of neurons over their preferred stimuli is technically challenging, there are cases where this assumption seems to hold. For example, research on the "oblique effect" supports the idea that the distribution of orientation tuning curves in V1 is proportional to the prior. Electrophysiology [13], optical imaging [14] and

fMRI studies [15] have found that there are more V1 neurons tuned to cardinal orientations than to oblique orientations. These findings are in agreement with the prior distribution of orientations of lines in the visual environment. Other evidence comes from motor areas. Repetitive stimulation of a finger expands its corresponding cortical representation in somatosensory area [16], suggesting more neurons are recruited to represent this stimulus. Alternatively, recruiting neurons $x_i^*$ according to the prior distribution can be implemented by modulating feature detection neurons' firing rates. This strategy also seems to be used by the brain: studies in parietal cortex [17] and superior colliculus [18] show that increased prior probability at a particular location results in stronger firing for neurons with receptive fields at that location.

Third, divisive normalization is a critical component in many neural models, notably in the study of attention modulation [19, 20]. It has been suggested that biophysical mechanisms such as shunting inhibition and synaptic depression might account for normalization and gain control [10, 21, 22]. Moreover, local interneurons [23] act as modulator for pooled inhibitory inputs and are good candidates for performing normalization. Our study makes no specific claims about the underlying biophysical processes, but gains support from the literature suggesting that there are plausible neural mechanisms for performing divisive normalization.

### 3.3 Importance sampling by Poisson spiking neurons

Neurons communicate mostly by spikes rather than continuous membrane potential signals. Poisson spiking neurons play an important role in other analyses of systems for representing probabilities [8]. Poisson spiking neurons can also be used to perform importance sampling if we have an ensemble of neurons with firing rates $\lambda_i$ proportional to $p(x|x_i^*)$, with the values of $x_i^*$ drawn from the prior. To show this we need a property of Poisson distributions: if $y_i \sim \text{Poisson}(\lambda_i)$ and $Y = \sum_i y_i$, then $Y \sim \text{Poisson}(\sum_i \lambda_i)$ and $(y_1, y_2, \ldots, y_m | Y = n) \sim \text{Multinomial}(n, \lambda_i / \sum_i \lambda_i)$. This further implies that $E(y_i/Y|Y = n) = \lambda_i / \sum_i \lambda_i$. Assume a neuron tuned to stimulus $x_i^*$ emits spikes $r_i \sim \text{Poisson}(c \cdot p(x|x_i^*))$, where $c$ is any positive constant. An average of a function $f(x_i^*)$ using the number of spikes produced by the corresponding neurons yields $\sum_i f(x_i^*) r_i / \sum_i r_i$, whose expectation is

$$E\left[\sum_i f(x_i^*) \frac{r_i}{\sum_j r_j}\right] = \sum_i f(x_i^*) E\left[\frac{r_i}{\sum_j r_j}\right] = \sum_i f(x_i^*) \frac{c\lambda_i}{\sum_j c\lambda_j} = \frac{\sum_i f(x_i^*) p(x|x_i^*)}{\sum_i p(x|x_i^*)} \quad (4)$$

which is thus an unbiased estimator of the importance sampling approximation to the posterior expectation. The variance of this estimator decreases as population activity $n = \sum_i r_i$ increases because $\text{var}[r_i/n] \sim 1/n$. Thus, Poisson spiking neurons, if plugged into an RBF network, can perform importance sampling and give similar results to "neurons" with analog output, as we confirm later in the paper through simulations.

## 4 Hierarchical Bayesian inference and multi-layer importance sampling

Inference tasks solved by the brain often involve more than one random variable, with complex dependency structures between those variables. For example, visual information process in primates involves dozens of subcortical areas that interconnect in a hierarchical structure containing two major pathways [5]. Hierarchical Bayesian inference has been proposed as a solution to this problem, with particle filtering and belief propagation as possible algorithms implemented by the brain [6]. However, few studies have proposed neural models that are capable of performing hierarchical Bayesian inference (although see [24]). We show how a multi-layer neural network can perform such computations using importance samplers (Fig. 1) as building blocks.

### 4.1 Generative models and Hierarchical Bayesian inference

Generative models describe the causal process by which data are generated, assigning a probability distribution to each step in that process. To understand brain function, it is often helpful to identify the generative model that determines how stimuli to the brain $S_x$ are generated. The brain then has to reverse the generative model to recover the latent variables expressed in the data (see Fig. 2). The direction of inference is thus the opposite of the direction in which the data are generated.

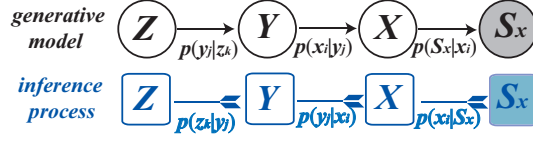

Figure 2: A hierarchical Bayesian model. The generative model specifies how each variable is generated (in circles), while inference reverses this process (in boxes). $S_x$ is the stimulus presented to the nervous system, while $X$, $Y$, and $Z$ are latent variables at increasing levels of abstraction.

In the case of a hierarchical Bayesian model, as shown in Fig. 2, the quantity of interest is the posterior expectation of some function $f(z)$ of a high-level latent variable $Z$ given stimulus $S_x$, $E[f(z)|S_x] = \int f(z)p(z|S_x)\,dz$. After repeatedly using the importance sampling trick (see Eq. 5), this hierarchical Bayesian inference problem can decomposed into three importance samplers with values $x_i^*$, $y_j^*$ and $z_k^*$ drawn from the prior.

$$
\begin{aligned}
E[f(z)|S_x] &= \int f(z)\,p(z|y)\left(\int p(y|x)p(x|S_x)\,dx\right)dy\,dz \quad\text{(importance sampling)} \quad x_i^* \sim p(x)\\
&\approx \int f(z)\,p(z|y)\,\frac{\sum_i p(y|x_i^*)p(S_x|x_i^*)}{\sum_i p(S_x|x_i^*)}\,dy\,dz\\
&= \int f(z)\,\frac{\sum_i \left(\int p(z|y)p(y|x_i^*)\,dy\right)p(S_x|x_i^*)}{\sum_i p(S_x|x_i^*)}\,dz \quad\text{(importance sampling)} \quad y_j^* \sim p(y)\\
&\approx \int f(z)\sum_i \frac{\sum_j p(z|y_j^*)p(x_i^*|y_j^*)}{\sum_j p(x_i^*|y_j^*)}\,\frac{p(S_x|x_i^*)}{\sum_i p(S_x|x_i^*)}\,dz\\
&= \sum_j \left(\int f(z)p(z|y_j^*)\,dz\right)\sum_i \frac{p(x_i^*|y_j^*)}{\sum_j p(x_i^*|y_j^*)}\,\frac{p(S_x|x_i^*)}{\sum_i p(S_x|x_i^*)} \quad\text{(importance sampling)} \quad z_k^* \sim p(z)\\
&\approx \sum_j \frac{\sum_k f(z_k^*)p(y_j^*|z_k^*)}{\sum_k p(y_j^*|z_k^*)}\sum_i \frac{p(x_i^*|y_j^*)}{\sum_j p(x_i^*|y_j^*)}\,\frac{p(S_x|x_i^*)}{\sum_i p(S_x|x_i^*)}\\
&= \sum_k f(z_k^*)\left(\sum_j \frac{p(y_j^*|z_k^*)}{\sum_k p(y_j^*|z_k^*)}\left(\sum_i \frac{p(x_i^*|y_j^*)}{\sum_j p(x_i^*|y_j^*)}\left(\frac{p(S_x|x_i^*)}{\sum_i p(S_x|x_i^*)}\right)\right)\right)\\
&\qquad\qquad\qquad\qquad z_k \qquad\qquad\qquad\qquad y_j \qquad\qquad\qquad x_i
\end{aligned}
\tag{5}
$$

This result relies on recursively applying importance sampling to the integral, with each recursion resulting in an approximation to the posterior distribution of another random variable. This recursive importance sampling scheme can be used in a variety of graphical models. For example, tracking a stimulus over time is a natural extension where an additional observation is added at each level of the generative model. We evaluate this scheme in several generative models in Section 5.

## 4.2 Neural implementation of the multi-layer importance sampler

The decomposition of hierarchical inference into recursive importance sampling (Eq. 5) gives rise to a multi-layer neural network implementation (see Fig. 3a). The input layer $X$ is similar to that in Fig. 1, composed of feature detection neurons with output proportional to the likelihood $p(S_x|x_i^*)$. Their output, after presynaptic normalization, is fed into a layer corresponding to the $Y$ variables, with synaptic weights $\frac{p(x_i^*|y_j^*)}{\sum_j p(x_i^*|y_j^*)}$. The response of neuron $y_j^*$, summing over synaptic inputs, approximates $p(y_j^*|S_x)$. Similarly, the response of $z_k^* \approx p(z_k^*|S_x)$, and the activities of these neurons are pooled to compute $E[f(z)|S_x]$. Note that, at each level, $x_i^*$, $y_j^*$ and $z_k^*$ are sampled from prior distributions. Posterior expectations involving any random variable can be computed because the neuron activities at each level approximate the posterior density. A single pool of neurons can also feed activation to multiple higher levels. Using the visual system as an example (Fig. 3b), such a multi-layer importance sampling scheme could be used to account for hierarchical inference in divergent pathways by projecting a set of V2 cells to both MT and V4 areas with corresponding synaptic weights.

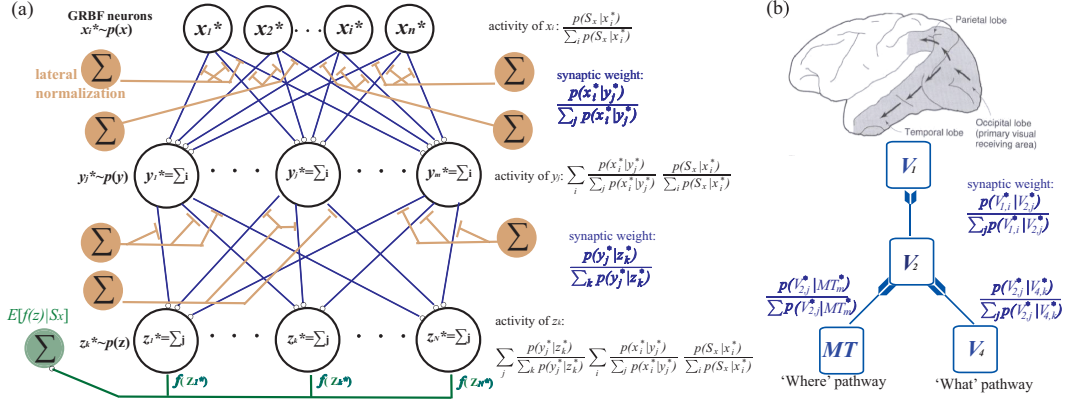

Figure 3: a) Multi-layer importance sampler for hierarchical Bayesian inference. b) Possible implementation in dorsal-ventral visual inference pathways, with multiple higher levels receiving input from one lower level. Note that the arrow directions in the figure are direction of inference, which is opposite to that of its generative model.

## 5 Simulations

In this section we examine how well the mechanisms introduced in the previous sections account for human behavioral data for two perceptual phenomena: cue combination and the oblique effect.

### 5.1 Haptic-visual cue combination

When sensory cues come from multiple modalities, the nervous system is able to combine those cues optimally in the way dictated by Bayesian statistics [2]. Fig. 4a shows the setup of an experiment where a subject measures the height of a bar through haptic and visual inputs. The object's visual input is manipulated so that the visual cues can be inconsistent with haptic cues and visual noise can be adjusted to different levels, i.e. visual cue follows $x_V \sim \mathcal{N}(S_V, \sigma_V^2)$ and haptic cue follows $x_H \sim \mathcal{N}(S_H, \sigma_H^2)$, where $S_V, S_H, \sigma_V^2$ are controlled parameters. The upper panel of Fig. 4d shows the percentage of trials that participants report the comparison stimulus (consistent visual/haptic cues from 45-65mm) is larger than the standard stimulus (inconsistent visual/haptic cues, $S_V = 60mm$ and $S_H = 50mm$). With the increase of visual noise, haptic input accounts for larger weights in decision making and the percentage curve is shifted towards $S_H$, consistent with Bayesian statistics.

Several studies have suggested that this form of cue combination could be implemented by population coding [2, 8]. In particular, [8] made an interesting observation that, for Poisson-like spiking neurons, summing firing activities of two populations is the optimal strategy. This model is under the Bayesian decoding framework and requires construction of the network so that these two populations of neurons have exactly the same number of neurons and precise one-to-one connection between two populations, with the connected pair of neurons having exactly the same tuning curves. We present an alternative solution based on importance sampling that encodes the probability distribution by a population of neurons directly.

The importance sampling solution approximates the posterior expectation of the bar's height $x_C^*$ given $S_V$ and $S_H$. Sensory inputs are channeled in through $x_V$ and $x_H$ (Fig.4b). Because sensory input varies in a small range (45-65mm in [2]), we assume priors $p(x_C)$, $p(x_V)$ and $p(x_H)$ are uniform. It is straightforward to approximate posterior $p(x_V|S_V)$ using importance sampling:

$$p(x_V = x_V^*|S_V) = E[\mathbf{1}(x_V = x_V^*)|S_V] \approx \frac{p(S_V|x_V^*)}{\sum_i p(S_V|x_{V,i}^*)} \approx \frac{r_V}{\sum_i r_{V,i}} \qquad x_{V,i}^* \sim p(x_V) \quad (6)$$

where $r_{V,i} \sim \text{Poisson}[c \cdot p(S_V|x_{V,i}^*)]$ is the number of spikes emitted by neuron $x_{V,i}^*$. A similar strategy applies to $p(x_H|S_H)$. The posterior $p(x_C|S_V, S_H)$, however, is not trivial since multiplication

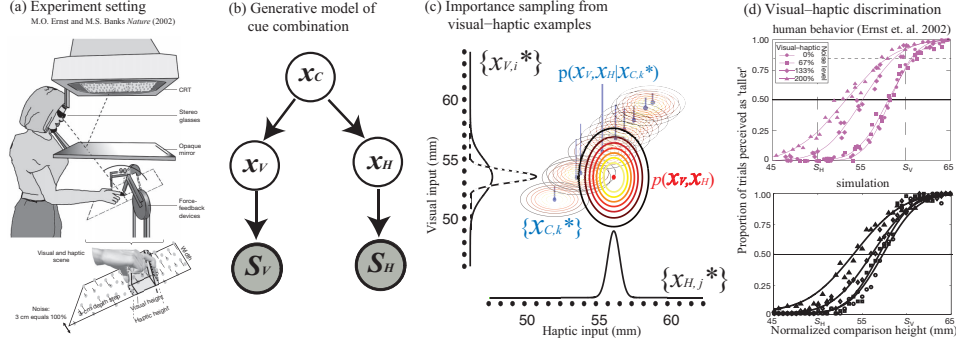

Figure 4: (a) Experimental setup [2]. (b) Generative model. $S_V$ and $S_H$ are the sensory stimuli, $X_V$ and $X_H$ the values along the visual and haptic dimensions, and $X_C$ the combined estimate of object height. (c) Illustration of importance sampling using two sensory arrays $\{x^*_{V,i}\}, \{x^*_{H,j}\}$. The transparent ellipses indicate the tuning curves of high level neurons centered on values $x^*_{C,k}$ over $x_V$ and $x_H$. The big ellipse represents the manipulated input with inconsistent sensory input and different variance structure. Bars at the center of opaque ellipses indicate the relative firing rates of $x_C$ neurons, proportional to $p(x^*_{C,k}|S_V, S_H)$. (d) Human data and simulation results.

of spike trains is needed.

$$
\begin{aligned}
p(x_C = x^*_C | S_V, S_H) &= \int \mathbf{1}(x_C = x^*_C)p(x_C|x_V, x_H)p(x_V|S_V)p(x_H|S_H)\, dx_V\, dx_H \\
&\approx \sum_i \sum_j \int \mathbf{1}(x_C = x^*_C)p(x_C|x_V, x_H)\frac{r_{V,i}}{\sum_i r_{V,i}}\frac{r_{H,j}}{\sum_j r_{H,j}} \quad (7)
\end{aligned}
$$

Fortunately, the experiment gives an important constraint, namely subjects were not aware of the manipulation of visual input. Thus, the values $x^*_{C,k}$ employed in the computation are sampled from normal perceptual conditions, namely consistent visual and haptic inputs ($x_V = x_H$) and normal variance structure (transparent ellipses in Fig.4c, on the diagonal). Therefore, the random variables $\{x_V, x_H\}$ effectively become one variable $x_{V,H}$ and values of $x^*_{V,H,i}$ are composed of samples drawn from $x_V$ and $x_H$ independently. Applying importance sampling,

$$
p(x_C = x^*_C | S_V, S_H) \approx \frac{\sum_i p(x^*_{V,i}|x^*_C)r_{V,i} + \sum_j p(x^*_{H,j}|x^*_C)r_{H,j}}{\sum_i r_{V,i} + \sum_j r_{H,j}} \quad (8)
$$

$$
E[x^*_C | S_V, S_H] \approx \sum_k x^*_{C,k} r_{C,k} / \sum_k r_{C,k} \quad (9)
$$

where $r_{C,k} \sim \text{Poisson}(c \cdot p(x^*_{C,k}|S_V, S_H))$ and $x^*_{C,k} \sim p(x_C)$. Compared with Eq. 6, inputs $x^*_{V,i}$ and $x^*_{H,j}$ are treaded as from one population in Eq 8. $r_{V,i}$ and $r_{H,j}$ are weighted differently only because of different observation noise. Eq. 9 is applicable for manipulated sensory input (in Fig. 4c, the ellipse off the diagonal). The simulation results (for an average of 500 trials) are shown in the lower panel of Fig.4d, compared with human data in the upper panel. There are two parameters, noise levels $\sigma_V$ and $\sigma_H$, are optimized to fit within-modality discrimination data (see [2] Fig. 3a). $\{x^*_{V,i}\}, \{x^*_{H,j}\}$ and $\{x^*_{C,k}\}$ consist of 20 independently drawn examples each, and the total firing rate of each set of neurons is limited to 30. The simulations produce a close match to human behavior.

## 5.2 The oblique effect

The oblique effect describes the phenomenon that people show greater sensitivity to bars with horizontal or vertical ($0^o/90^o$) orientations than "oblique" orientations. Fig. 5a shows an experimental setup where subjects exhibited higher sensitivity in detecting the direction of rotation of a bar when the reference bar to which it was compared was in one of these cardinal orientations. Fig. 5b shows the generative model for this detection problem. The top-level binary variable $D$ randomly chooses a direction of rotation. Conditioning on $D$, the amplitude of rotation $\Delta\theta$ is generated from a truncated

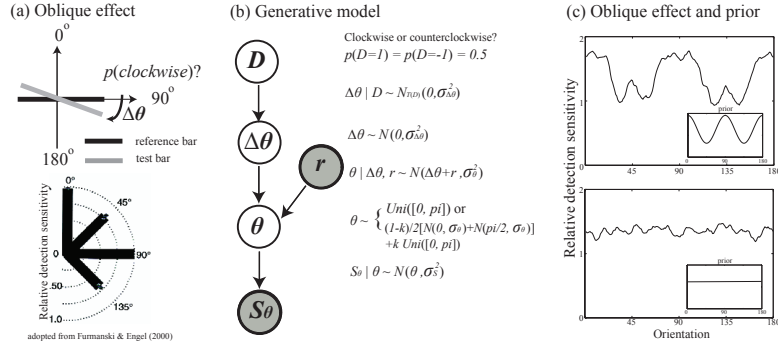

Figure 5: (a) Orientation detection experiment. The oblique effect is shown in lower panel, being greater sensitivity to orientation near the cardinal directions. (b) Generative model. (c) The oblique effect emerges from our model, but depends on having the correct prior $p(\theta)$.

normal distribution ($N_{T(D)}$), being restricted to $\Delta\theta > 0$ if $D = 1$ and $\Delta\theta < 0$ otherwise). When combined with the angle of the reference bar $r$ (shaded in the graphical model, since it is known), $\Delta\theta$ generates the orientation of a test bar $\theta$, and $\theta$ further generates the observation $S_\theta$, both with normal distributions with variance $\sigma_\theta$ and $\sigma_{S_\theta}$ respectively.

The oblique effect has been shown to be closely related to the number of V1 neurons that tuned to different orientations [25]. Many studies have found more V1 neurons tuned to cardinal orientations than other orientations [13, 14, 15]. Moreover, the uneven distribution of feature detection neurons is consistent with the idea that these neurons might be sampled proportional to the prior: more horizontal and vertical segments exist in the natural visual environment of humans.

Importance sampling provides a direct test of the hypothesis that preferential distribution of V1 neurons around $0^o/90^o$ can cause the oblique effect, which becomes a question of whether the oblique effect depends on the use of a prior $p(\theta)$ with this distribution. The quantity of interest is:

$$p(D = 1|S_\theta, r) \approx \sum_{j'} \sum_i \frac{p(\theta_i^*|\Delta\theta_{j'}^*, r)}{\sum_j p(\theta_i^*|\Delta\theta_j^*, r)} \frac{p(S_\theta|\theta_i^*)}{\sum_i p(S_\theta|\theta_i^*)} \tag{10}$$

where $j'$ indexes all $\Delta\theta^* > 0$. If $p(D = 1|S_\theta, r) > 0.5$, then we should assign $D = 1$. Fig. 5c shows that detection sensitivity is uncorrelated with orientations if we take a uniform prior $p(\theta)$, but exhibits the oblique effect under a prior that prefers cardinal directions. In both cases, 40 neurons are used to represent each of $\Delta\theta_i^*$ and $\theta_i^*$, and results are averaged over 100 trials. Sensitivity is measured by percentage correct in inference. Due to the qualitative nature of this simulation, model parameters are not tuned to fit experiment data.

## 6   Conclusion

Understanding how the brain solves the problem of Bayesian inference is a significant challenge for computational neuroscience. In this paper, we have explored the potential of a class of solutions that draw on ideas from computer science, statistics, and psychology. We have shown that a small number of feature detection neurons whose tuning curves represent a small set of typical examples from sensory experience is sufficient to perform some basic forms of Bayesian inference. Moreover, our theoretical analysis shows that this mechanism corresponds to a Monte Carlo sampling method, i.e. importance sampling. The basic idea behind this approach – storing examples and activating them based on similarity – is at the heart of a variety of psychological models, and straightforward to implement either in traditional neural network architectures like radial basis function networks, circuits of Poisson spiking neurons, or associative memory models. The nervous system is constantly reorganizing to capture the ever-changing structure of our environment. Components of the importance sampler, such as the tuning curves and their synaptic strengths, need to be updated to match the distributions in the environment. Understanding how the brain might solve this daunting problem is a key question for future research.

**Acknowledgments.** Supported by the Air Force Office of Scientific Research (grant FA9550-07-1-0351).

# References

[1] K. Körding and D. M. Wolpert. Bayesian integration in sensorimotor learning. *Nature*, 427:244–247, 2004.

[2] M. O. Ernst and M. S. Banks. Humans integrate visual and haptic information in a statistically optimal fashion. *Nature*, 415(6870):429–433, 2002.

[3] A. Stocker and E. Simoncelli. A bayesian model of conditioned perception. In J.C. Platt, D. Koller, Y. Singer, and S. Roweis, editors, *Advances in Neural Information Processing Systems 20*, pages 1409–1416. MIT Press, Cambridge, MA, 2008.

[4] A. P. Blaisdell, K. Sawa, K. J. Leising, and M. R. Waldmann. Causal reasoning in rats. *Science*, 311(5763):1020–1022, 2006.

[5] D. C. Van Essen, C. H. Anderson, and D. J. Felleman. Information processing in the primate visual system: an integrated systems perspective. *Science*, 255(5043):419–423, 1992 Jan 24.

[6] T. S. Lee and D. Mumford. Hierarchical bayesian inference in the visual cortex. *J.Opt.Soc.Am.A Opt.Image Sci.Vis.*, 20(7):1434–1448, 2003.

[7] R. S. Zemel, P. Dayan, and A. Pouget. Probabilistic interpretation of population codes. *Neural Comput*, 10(2):403–430, 1998.

[8] W. J. Ma, J. M. Beck, P. E. Latham, and A. Pouget. Bayesian inference with probabilistic population codes. *Nat.Neurosci.*, 9(11):1432–1438, 2006.

[9] L. Shi, N. H. Feldman, and T. L. Griffiths. Performing bayesian inference with exemplar models. In *Proceedings of the 30th Annual Conference of the Cognitive Science Society*, 2008.

[10] M. Kouh and T. Poggio. A canonical neural circuit for cortical nonlinear operations. *Neural Comput*, 20(6):1427–1451, 2008.

[11] J. K. Kruschke. Alcove: An exemplar-based connectionist model of category learning. *Psychological Review*, 99:22–44, 1992.

[12] M. J. D. Powell. *Radial basis functions for multivariable interpolation: a review*. Clarendon Press, New York, NY, USA, 1987.

[13] R. L. De Valois, E. W. Yund, and N. Hepler. The orientation and direction selectivity of cells in macaque visual cortex. *Vision Res*, 22(5):531–544, 1982.

[14] D. M. Coppola, L. E. White, D. Fitzpatrick, and D. Purves. Unequal representation of cardinal and oblique contours in ferret visual cortex. *Proc Natl Acad Sci U S A*, 95(5):2621–2623, 1998 Mar 3.

[15] C. S. Furmanski and S. A. Engel. An oblique effect in human primary visual cortex. *Nat Neurosci*, 3(6):535–536, 2000.

[16] A. Hodzic, R. Veit, A. A. Karim, M. Erb, and B. Godde. Improvement and decline in tactile discrimination behavior after cortical plasticity induced by passive tactile coactivation. *J Neurosci*, 24(2):442–446, 2004.

[17] M. L. Platt and P. W. Glimcher. Neural correlates of decision variables in parietal cortex. *Nature*, 400:233–238, 1999.

[18] M. A. Basso and R. H. Wurtz. Modulation of neuronal activity by target uncertainty. *Nature*, 389(6646):66–69, 1997.

[19] J. H. Reynolds and D. J. Heeger. The normalization model of attention. *Neuron*, 61(2):168–185, 2009 Jan 29.

[20] J. Lee and J. H. R. Maunsell. A normalization model of attentional modulation of single unit responses. *PLoS ONE*, 4(2):e4651, 2009.

[21] S. J. Mitchell and R. A. Silver. Shunting inhibition modulates neuronal gain during synaptic excitation. *Neuron*, 38(3):433–445, 2003.

[22] J. S. Rothman, L. Cathala, V. Steuber, and R A. Silver. Synaptic depression enables neuronal gain control. *Nature*, 457(7232):1015–1018, 2009 Feb 19.

[23] H. Markram, M. Toledo-Rodriguez, Y. Wang, A. Gupta, G. Silberberg, and C. Wu. Interneurons of the neocortical inhibitory system. *Nat Rev Neurosci*, 5(10):793–807, 2004 Oct.

[24] K. Friston. Hierarchical models in the brain. *PLoS Comput Biol*, 4(11):e1000211, 2008 Nov.

[25] G. A. Orban, E. Vandenbussche, and R. Vogels. Human orientation discrimination tested with long stimuli. *Vision Res*, 24(2):121–128, 1984.

